# Simple Spin Models
# for the Development of Ocular Dominance
# Columns and Iso-0rientation Patches

**J.D. Cowan & A.E. Friedman**
Department of Mathematics, Committee on
Neurobiology, and Brain Research Institute,
The University of Chicago, 5734 S. Univ. Ave.,
Chicago, Illinois 60637

## Abstract

Simple classical spin models well-known to physicists as the ANNNI
and Heisenberg XY Models, in which long-range interactions occur in
a pattern given by the Mexican Hat operator, can generate many of the
structural properties characteristic of the ocular dominance columns
and iso-orientation patches seen in cat and primate visual cortex.

## 1 INTRODUCTION

In recent years numerous models for the formation of ocular dominance columns
(Malsburg, 1979 ; Swindale, 1980; Miller, Keller, & Stryker, 1989) and of iso-orientation
patches (Malsburg 1973; Swindale 1982 & Linsker 1986)have been published. Here we
show that simple spin models can reproduce many of the observed features. Our work is
similar to, but independent of a recent study employing spin models (Tanaka, 1990).

## 1.1 OCULAR DOMINANCE COLUMNS

We use a one-dimensional classical spin Hamiltonian on a two-dimensional lattice with long-range interactions. Let $\sigma_i$ be a spin vector restricted to the orientations $\uparrow$ and $\downarrow$ in the lattice space, and let the spin Hamiltonian be:

$$H_{OD} = -\sum_i \sum_{j \neq i} w_{ij}\, \sigma_i \cdot \sigma_j, \qquad (1)$$

where $w_{ij}$ is the well-known "Mexican Hat" distribution of weights:

$$w_{ij} = a_+ \exp(-|i\text{-}j|^2/\sigma_+^2) - a_- \exp(-|i\text{-}j|^2/\sigma_-^2) \qquad (2)$$

with $\sigma_+ < \sigma_-$ and $a_+/a_- = \sigma_-^2/\sigma_+^2$. Evidently $\sigma_i \cdot \sigma_j = \pm |\sigma_i||\sigma_j| = \pm 1$, so that

$$H_{OD} = -\sum_i \sum_{j \neq i} w_{ij}^s - \sum_i \sum_{j \neq i} w_{ij}^o \qquad (3)$$

where $w_{ij}^s = w_{ij}$ if $\sigma_i = \sigma_j$, and $w_{ij}^o = -w_{ij}$ if $\sigma_i \neq \sigma_j$.

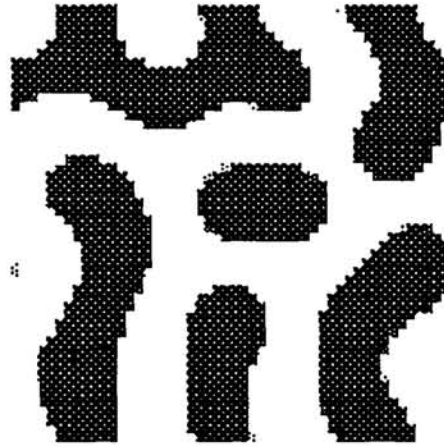

**Figure 1.** Pattern of Ocular Dominance which results from simulated annealing of the energy function $H_{OD}$. Light and dark shadings correspond respectively to the two eyes.

Let s denote retinal fibers from the same eye and o fibers from the opposite eye. Then $H_{OD}$ represents the "energy" of interactions between fibers from the two eyes. It is relatively easy to find a configuration of spins which minimizes $H_{OD}$ by simulated annealing (Kirkpatrick, Gelatt & Vecchi 1983). The result is shown in figure 1. It will be seen that the resulting pattern of right and left eye spins $\sigma^R$ and $\sigma^L$ is disordered, but at a constant wavelength determined in large part by the space constants $\sigma_+$ and $\sigma_-$.

Breaking the symmetry of the initial conditions (or letting the lattivce grow systematically) results in ordered patterns.

If $H_{OD}$ is considered to be the energy function of a network of spins exhibiting gradient dynamics (Hirsch & Smale, 1974), then one can write equations for the evolution of spin patterns in the form:

$$\frac{d}{dt} \sigma_i^\alpha = -\frac{\partial}{\partial \sigma_i^\alpha} H_{OD} = \sum_{j \neq i} w_{ij}^{\alpha\beta} \sigma_j^\beta$$

$$= \sum_{j \neq i} w_{ij}^s \sigma_i^\alpha + \sum_{j \neq i} w_{ij}^o \sigma_i^\beta = \sum_{j \neq i} w_{ij} \sigma_i^\alpha - \sum_{j \neq i} w_{ij} \sigma_i^\beta, \qquad (4)$$

where $\alpha$ = R or L, $\beta$ = L or R respectively. Equation (4) will be recognized as that proposed by Swindale in 1979.

## 1. 2 ISO-ORIENTATION PATCHES

Now let $\sigma_i$ represent avector in the plane of the lattice which runs continuously from $\uparrow$ to $\downarrow$ without reference to eye class. It follows that

$$\sigma_i \cdot \sigma_j = |\sigma_i| \, |\sigma_i| \cos(\theta_i - \theta_j) \qquad (5)$$

where $\theta_i$ is the orientation of the ith spin vector. The appropriate classical spin Hamiltonian is:

$$H_{IO} = -\sum_i \sum_{j \neq i} w_{ij} \sigma_i \cdot \sigma_j = -\sum_i \sum_{j \neq i} w_{ij} |\sigma_i| \, |\sigma_i| \cos(\theta_i - \theta_j). \qquad (6)$$

Physicists will recognize $H_{OD}$ as a form of the Ising Lattice Hamiltonian with long-range alternating next nearest neighbor interactions, a type of ANNNI model (Binder, 1986) and $H_{IO}$ as a similar form of the Heisenberg XY Model for antiferromagnetic materials (Binder 1986).

Again one can find a spin configuration that minimizes $H_{IO}$ by simulated annealing. The result is shown in figure 2 in which six differing orientations are depicted, corresponding to $30°$ increments (note that $\theta + \pi$ is equivalent to $\theta$). It will be seen that there are long stretches of continuously changing spin vector orientations, with intercalated discontinuities and both clockwise and counter-clockwise singular regions around which the orientations rotate. A one-dimensional slice shows some of these features, and is shown in figure 3.

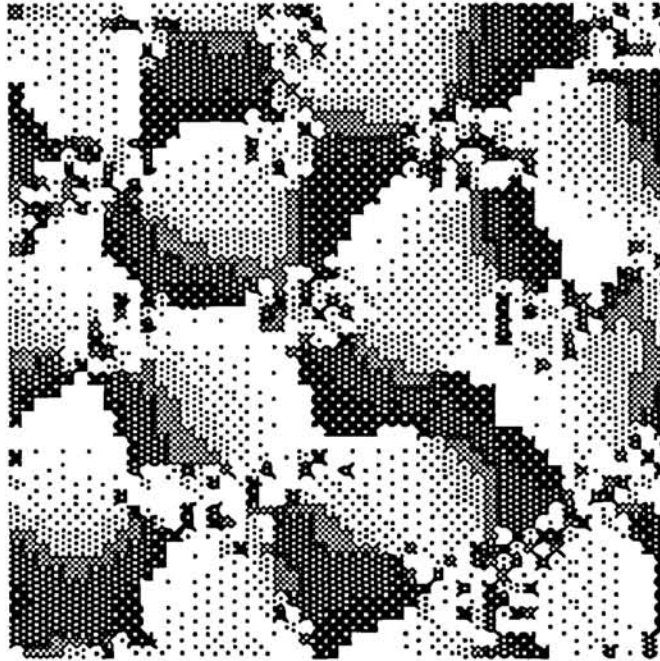

**Figure 2.** Pattern of orientation patches obtained by simulated annealing of the energy function $H_{IO}$. Six differing orientations varying from $0°$ to $180°$ are represented by the different shadings.

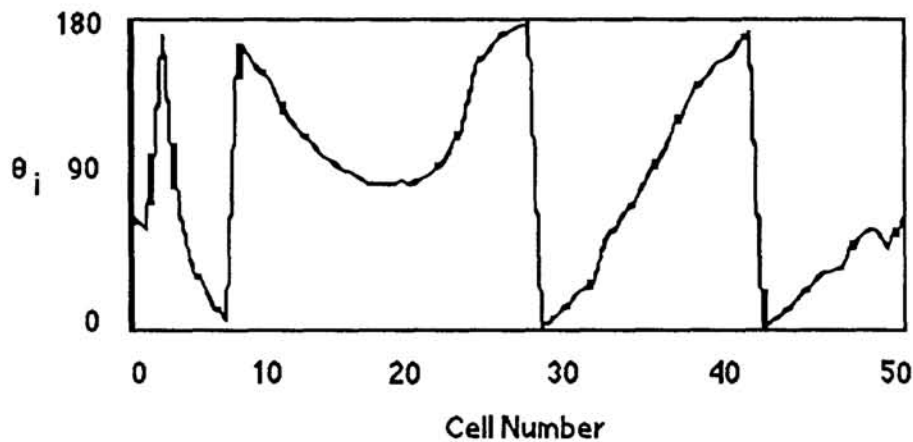

**Figure 3.** Details of a one-dimensional slice through the orientation map. Long stretches of smoothly changing orientations are evident.

The length of $\sigma_i$ is also correlated with these details. Figure 4 shows that $|\sigma_i|$ is large in smoothly changing regions and smallest in the neighborhood of a singularity. In fact this model reproduces most of the details of iso-orientation patches found by Blasdel and Salama (1986).

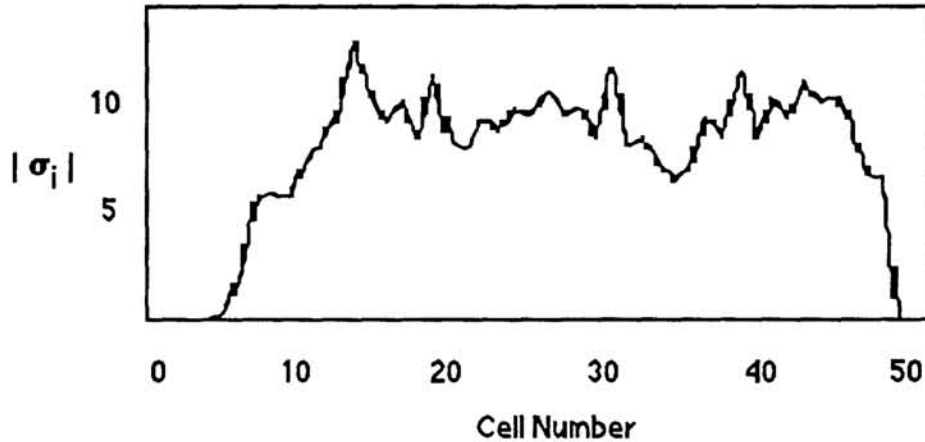

**Figure 4.** Variation of |$\sigma_i$| along the same one-dim. slice through the orientation map shown in figure 3. The amplitude drops only near singular regions.

For example, the change in orientation per unit length, |grad$\theta_i$| is shown in figure 5. It will be seen that the lattice is "tiled", just as in the data from visual cortex, with max |grad$\theta_i$| located at singularities.

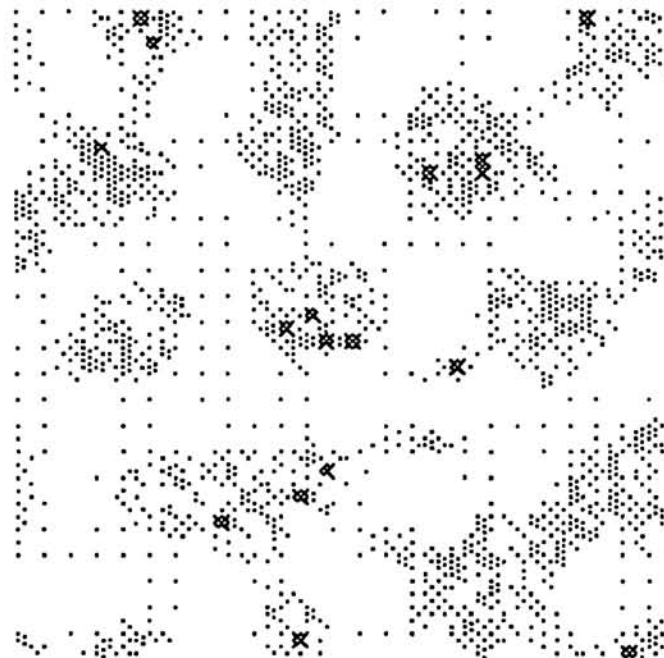

**Figure 5.** Plot of |grad$\theta_i$| corresponding to the orientation map of figure 2. Regions of maximum rate of change of $\theta_i$ are shown as shaded. These correspond with the singular regions of figure 2.

Once again, if $H_{IO}$ is taken to be the energy of a gradient dynamical system, there results the equation:

$$\frac{d}{dt} \sigma_i = -\frac{\partial}{\partial \sigma_i} H_{IO} = \sum_{j \neq i} w_{ij}\sigma_j \qquad (7)$$

which is exactly that equation introduced by Swindale in 1981 as a model for the structure of iso-orientation patches. There is an obvious relationship between such equations, and recent similar treatments (Durbin & Mitchison 1990; Schulten, K. 1990 (Preprint); Cherjnavsky & Moody, 1990).

## 2 CONCLUSIONS

Simple classical spin models well-known to physicists as the ANNNI and Heisenberg XY Models, in which long-range interactions occur in a pattern given by the Mexican Hat operator, can generate many of the structural properties characteristic of the ocular dominance columns and iso-orientation patches seen in cat and primate visual cortex.

### Acknowledgements

This work is based on lectures given at the Institute for Theoretical Physics (Santa Barbara) Workshop on Neural Networks and Spin Glasses, in 1986. We thank the Institute and The University of Chicago Brain Research Foundation for partial support of this work.

### References

Malsburg, Ch.v.d. (1979), Biol. Cybern., **32**, 49-62.
Swindale, N.V. (1980), Proc. Roy. Soc. Lond. B, **208**, 243-264.
Miller, K.D., Keller, J.B. & Stryker, M. P. (1989), Science, **245**, 605-611.
Malsburg, Ch.v.d. (1973), Biol. Cybern., **14**, 85-100.
Swindale, N.V. (1982), Proc. Roy. Soc. Lond. B, **215**, 211-230.
Linsker, R. (1986), PNAS, **83**, 7508-7512; 8390-8394; 8779-8783.
Tanaka, S. (1990), Neural Networks, **3**, 6, 625-640.
Kirkpatrick, S., Gelatt, C.D. Jr. & Vecchi, M.P. (1983), Science, **229**, 671-679.
Hirsch, M.W. & Smale, S. (1974), Differential Equations, Dynamical Systems, and Linear Algebra. (Academic Press, NY).
Binder, K. (1986), Monte Carlo Methods in Statistical Physics, (Springer, NY.).
Blasdel, G.G. & Salama, G. (1986), Nature, **321**, 579-587.
Durbin, R. & Mitchison, G. (1990), Nature, **343**, 6259, 644-647.
Schulten, K. (1990) (Preprint).
Cherjnavsky, A. & Moody, J. (1990), Neural Computation, **2**, 3, 334-354.
